# Recognizing Evoked Potentials in a Virtual Environment *

**Jessica D. Bayliss and Dana H. Ballard**
Department of Computer Science
University of Rochester
Rochester, NY 14627
{*bayliss,dana*}@*cs.rochester.edu*

## Abstract

Virtual reality (VR) provides immersive and controllable experimental environments. It expands the bounds of possible evoked potential (EP) experiments by providing complex, dynamic environments in order to study cognition without sacrificing environmental control. VR also serves as a safe dynamic testbed for brain-computer interface (BCI) research. However, there has been some concern about detecting EP signals in a complex VR environment. This paper shows that EPs exist at red, green, and yellow stop lights in a virtual driving environment. Experimental results show the existence of the P3 EP at "go" and "stop" lights and the contingent negative variation (CNV) EP at "slow down" lights. In order to test the feasibility of on-line recognition in VR, we looked at recognizing the P3 EP at red stop lights and the absence of this signal at yellow slow down lights. Recognition results show that the P3 may successfully be used to control the brakes of a VR car at stop lights.

## 1 Introduction

The controllability of VR makes it an excellent candidate for use in studying cognition. It expands the bounds of possible evoked potential (EP) experiments by providing complex, dynamic environments in order to study decision making in cognition without sacrificing environmental control. We have created a flexible system for real-time EEG collection and analysis from within virtual environments.

The ability of our system to give quick feedback enables it to be used in brain-computer interface (BCI) research, which is aimed at helping individuals with severe motor deficits to become more independent. Recent BCI work has shown the feasibility of on-line averaging and biofeedback methods in order to choose characters or move a cursor on a computer screen with up to 95% accuracy while sitting still and concentrating on the screen [McFarland *et al.*, 1993; Pfurtscheller *et al.*, 1996; Vaughn *et al.*, 1996; Farwell and Donchin, 1988]. Our focus is to dramatically extend the BCI by allowing evoked potentials to propel the user through alternate virtual environments. For example, a

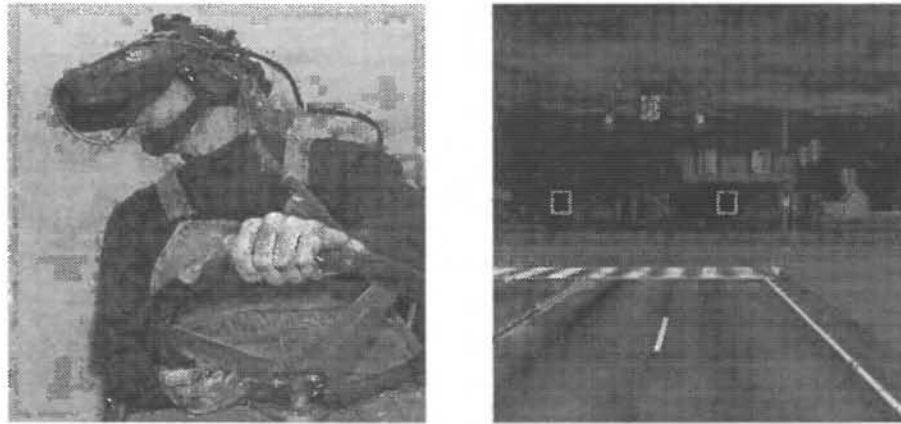

Figure 1: (Left) An individual demonstrates driving in the modified go cart. (Right) A typical stoplight scene in the virtual environment.

user could choose a virtual living room from a menu of rooms, navigate to the living room automatically in the head-mounted display, and then choose to turn on the stereo.

As shown in [Farwell and Donchin, 1988], the P3 EP may be used for a brain-computer interface that picks characters on a computer monitor. Discovered by [Chapman and Bragdon, 1964; Sutton *et al.*, 1965] and extensively studied (see [Polich, 1998] for a literature review), the P3 is a positive waveform occurring approximately 300-500 ms after an infrequent task-relevant stimulus. We show that requiring subjects to stop or go at virtual traffic lights elicits this EP. The contingent negative variation (CNV), an EP that happens preceding an expected stimulus, occurs at slow down lights.

In order to test the feasibility of on-line recognition in the noisy VR environment, we recognized the P3 EP at red stop lights and the lack of this signal at yellow slow down lights. Results using a robust Kalman filter for off-line recognition indicate that the car may be stopped reliably with an average accuracy of 84.5% while the on-line average for car halting is 83%.

## 2   The Stoplight Experiments

The first experiment we performed in the virtual driving environment shows that a P3 EP is obtained when subjects stop or go at a virtual light and that a CNV occurs when subjects see a slow down light. Since all subjects received the same light colors for the slow down, go, and stop conditions we then performed a second experiment with different light colors in order to disambiguate light color from the occurrence of the P3 and CNV.

Previous P3 research has concentrated primarily on static environments such as the continuous performance task [Rosvold *et al.*, 1956]. In the visual continuous performance task (VCPT), static images are flashed on a screen and the subject is told to press a button when a rare stimulus occurs or to count the number of occurrences of a rare stimulus. This makes the stimulus both rare and task relevant in order to evoke a P3. As an example, given red and yellow stoplight pictures, a P3 should occur if the red picture is less frequent than the yellow and subjects are told to press a mouse button only during the red light. We assumed a similar response would occur in a VR driving world if certain lights were infrequent and subjects were told to stop or go at them. This differs from the VCPT in two important ways:

1. In the VCPT subjects sit passively and respond to stimuli. In the driving task,

subjects control when the stimuli appear by where they drive in the virtual world.

2. Since subjects are actively involved and fully immersed in the virtual world, they make more eye and head movements. The movement amount can be reduced by a particular experimental paradigm, but it can not be eliminated.

The first difference makes the VR environment a more natural experimental environment. The second difference means that subjects create more data artifacts with extra movement. We handled these artifacts by first manipulating the experimental environment to reduce movements where important stimulus events occurred. This meant that all stoplights were placed at the end of straight stretches of road in order to avoid the artifacts caused by turning a corner. For our on-line recognition, we then used the eye movement reduction technique described in [Semlitsch *et al.*, 1986] in order to subtract a combination of the remaining eye and head movement artifact.

## 2.1 Experimental Setup

All subjects used a modified go cart in order to control the virtual car (see Figure 1). The virtual reality interface is rendered on a Silicon Graphics Onyx machine with 4 processors and an Infinite Reality Graphics Engine. The environment is presented to the subject through a head-mounted display (HMD). Since scalp EEG recordings are measured in microvolts, electrical signals may easily interfere during an experiment. We tested the effects of wearing a VR4 HMD containing an ISCAN eye tracker and discovered that the noise levels inside of the VR helmet were comparable to noise levels while watching a laptop screen [Bayliss and Ballard, 1998].

A trigger pulse containing information about the color of the light was sent to the EEG acquisition system whenever a light changed. While an epoch size from -100 ms to 1 sec was specified, the data was recorded continuously. Information about head position as well as gas, braking, and steering position were saved to an external file. Eight electrodes sites (FZ, CZ, CPZ, PZ, P3, P4, as well as 2 vertical EOG channels) were arranged on the heads of seven subjects with a linked mastoid reference. Electrode impedances were between 2 and 5 kohms for all subjects. Subjects ranged in age from 19 to 52 and most had no previous experiences in a virtual environment. The EEG signal was amplified using Grass amplifiers with an analog bandwidth from 0.1 to 100 Hz. Signals were then digitized at a rate of 500 Hz and stored to a computer.

## 2.2 Ordinary Traffic Light Color Experiment

Five subjects were instructed to slow down on yellow lights, stop for red lights, and go for green lights. These are normal traffic light colors. Subjects were allowed to drive in the environment before the experiment to get used to driving in VR.

In order to make slow down lights more frequent, all stoplights turned to the slow down color when subjects were further than 30 meters aways from them. When the subject drove closer than 30 meters the light then turned to either the go or stop color with equal probability. The rest of the light sequence followed normal stoplights with the stop light turning to the go light after 3 seconds and the go light not changing.

We calculated the grand averages over red, green, and yellow light trials (see Figure 2a). Epochs affected by artifact were ignored in the averages in order to make sure that any existing movements were not causing a P3-like signal. Results show that a P3 EP occurs for both red and green lights. Back averaging from the green/red lights to the yellow light shows the existence of a CNV starting at approximately 2 seconds before the light changes to red or green.

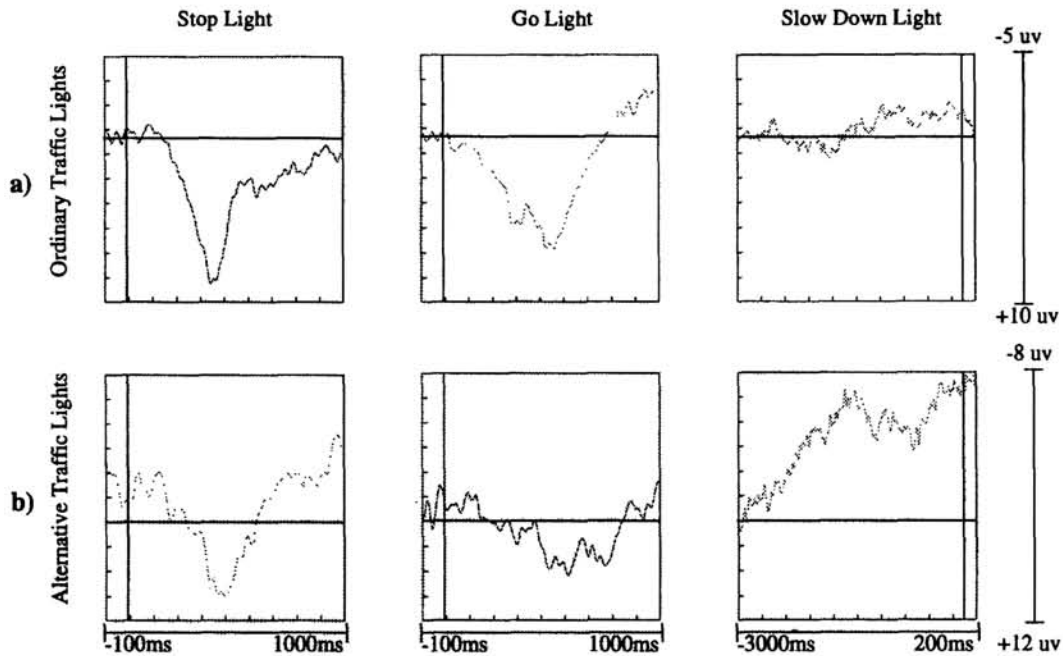

Figure 2: a) Grand averages for the red stop, green go, and yellow slow down lights. b) Grand averages for the yellow stop, red go, and green slow down lights. All slow down lights have been back-averaged from the occurrence of the go/stop light in order to show the existence of a CNV.

## 2.3  Alternative Traffic Light Colors

The P3 is related to task relevance and should not be related to color, but color needed to be disambiguated as the source of the P3 in the experiment. We had two subjects slow down at green lights, stop at yellow lights, and go at red lights. In order to get used to this combination of colors, subjects were allowed to drive in the town before the experiment.

The grand averages for each light color were calculated in the same manner as the averages above and are shown in Figure 2b. As expected, a P3 signal existed for the stop condition and a CNV for the slow down condition. The go condition P3 was much noisier for these two subjects, although a slight P3-like signal is still visible.

## 3  Single Trial Recognition Results

While averages show the existence of the P3 EP at red stop lights and the absence of such at yellow slow down lights, we needed to discover if the signal was clean enough for single trial recognition as the quick feedback needed by a BCI depends on quick recognition. While there were three light conditions to recognize, there were only two distinct kinds of evoked potentials. We chose to recognize the difference between the P3 and the CNV since their averages are very different. Recognizing the difference between two kinds of EPs gives us the ability to use a BCI in any task that can be performed using a series of binary decisions. We tried three methods for classification of the P3 EP: correlation, independent component analysis (ICA), and a robust Kalman filter.

Approximately, 90 slow down yellow light and 45 stop red light trials from each subject were classified. The reason we allowed a yellow light bias to enter recognition is because the yellow light currently represents an unimportant event in the environment. In a real BCI unimportant events are likely to occur more than user-directed actions, making this bias justifiable.

Table 1: Recognition Results ($p < 0.01$)

| Subjects | Correlation %Correct | | | ICA %Correct | | | Robust Kalman Filter %Correct | | |
|---|---|---|---|---|---|---|---|---|---|
| | Red | Yel | Total | Red | Yel | Total | Red | Yel | Total |
| S1 | 81 | 51 | 64 | 76 | 77 | 77 | 55 | 86 | 77 |
| S2 | 95 | 63 | 73 | 86 | 88 | 87 | 82 | 94 | 90 |
| S3 | 89 | 56 | 66 | 72 | 87 | 82 | 74 | 85 | 81 |
| S4 | 81 | 60 | 67 | 73 | 69 | 71 | 65 | 91 | 82 |
| S5 | 63 | 66 | 65 | 65 | 79 | 74 | 78 | 92 | 87 |

Table 2: Recognition Results for Return Subjects

| Subjects | Robust K-Filter % Correct | | |
|---|---|---|---|
| | Red | Yel | Total |
| S4 | 73 | 90 | 85 |
| S5 | 67 | 87 | 80 |

As expected, the data obtained while driving contained artifacts, but in an on-line BCI these artifacts must be reduced in order to make sure that what the recognition algorithm is recognizing is not an artifact such as eye movement. In order to reduce these artifacts, we performed the on-line linear regression technique described in [Semlitsch *et al.*, 1986] in order to subtract a combination of eye and head movement artifact.

In order to create a baseline from which to compare the performance of other algorithms, we calculated the correlation of all sample trials with the red and yellow light averages from each subject's maximal P3 electrode site using the following formula:

$$\textbf{correlation} =$$
$$(\textbf{sample} * \textbf{ave}^T)/(\| \textbf{sample} \| * \| \textbf{ave} \|) \tag{1}$$

where **sample** and **ave** are both $1 \times 500$ vectors representing the trial epochs and light averages (respectively). We used the whole trial epoch for recognition because it yielded better recognition than just the time area around the P3. If the highest correlation of a trial epoch with the red and yellow averages was greater than 0.0, then the signal was classified as that type of signal. If both averages correlated negatively with the single trial, then the trial was counted as a yellow light signal. As can be seen in Table 1, the correct signal identification of red lights was extremely high while the yellow light identification pulled the results down. This may be explained by the greater variance of the yellow light epochs. Correlations in general were poor with typical correlations around 0.25.

ICA has successfully been used in order to minimize artifacts in EEG data [Jung *et al.*, 1997; Vigario, 1997] and has also proven useful in separating P3 component data from an averaged waveform [Makeig *et al.*, 1997]. The next experiment used ICA in order to try to separate the background EEG signal from the P3 signal. Independent component analysis (ICA) assumes that $n$ EEG data channels x are a linear combination of $n$ statistically *independent* signals s:

$$\mathbf{x} = A\mathbf{s} \tag{2}$$

where x and s are $n \times 1$ vectors. We used the matlab package mentioned in [Makeig *et al.*, 1997] with default learning values, which finds a matrix $W$ by stochastic gradient descent.

This matrix $W$ performs component separation. All data was sphered in order to speed convergence time.

After training the $W$ matrix, the source channel showing the closest P3-like signal (using correlation with the average) for the red light average data was chosen as the signal with which to correlate individual epochs. The trained $W$ matrix was also used to find the sources of the yellow light average. The red and yellow light responses were then correlated with individual epoch sources in the manner of the first experiment.

The third experiment used the robust Kalman filter framework formulated by Rao [Rao, 1998]. The Kalman filter assumes a linear model similar to the one of ICA in equation 2, but assumes the EEG output $\mathbf{x}$ is the observable output of a generative or measurement matrix $A$ and an internal state vector $\mathbf{s}$ of *Gaussian* sources. The output may also have an additional noise component $\mathbf{n}$, a Gaussian stochastic noise process with mean zero and a covariance matrix given by $\Sigma = E[\mathbf{n}\mathbf{n}^{\mathbf{T}}]$, leading to the model expression: $\mathbf{x} = A\mathbf{s} + \mathbf{n}$. In order to find the most optimal value of $\mathbf{s}$, a weighted least-squares criterion is formulated:

$$ J = (\mathbf{x} - A\mathbf{s})^{T}\Sigma^{-1}(\mathbf{x} - A\mathbf{s}) + (\mathbf{s} - \bar{\mathbf{s}})^{T}M^{-1}(\mathbf{s} - \bar{\mathbf{s}}) \qquad (3) $$

where $\mathbf{s}$ follows a Gaussian distribution with mean $\bar{\mathbf{s}}$ and covariance $M$. Minimizing this criterion by setting $\frac{\partial J}{\partial \mathbf{s}} = 0$ and using the substitution $N = (A^{T}\Sigma^{-1}U + M^{-1})^{-1}$ yields the Kalman filter equation, which is basically equal to the old estimate plus the Kalman gain times the residual error.

$$ \mathbf{s} = \bar{\mathbf{s}} + NA^{T}\Sigma^{-1}(\mathbf{x} - A\bar{\mathbf{s}}) \qquad (4) $$

In an analogous manner, the measurement matrix $A$ may be estimated (learned) if one assumes the physical relationships encoded by the measurement matrix are relatively stable. The learning rule for the measurement matrix may be derived in a manner similar to the rule for the internal state vector. In addition, a decay term is often needed in order to avoid overfitting the data set. See [Rao, 1998] for details.

In our experiments both the internal state matrix $\mathbf{s}$ and the measurement matrix $A$ were learned by training them on the average red light signal and the average yellow light signal. The signal is measured from the start of the trial which is known since it is triggered by the light change. We used a Kalman gain of 0.6 and a decay of 0.3. After training, the signal estimate for each epoch is correlated with the red and yellow light signal estimates in the manner of experiment 1. We made the Kalman filter statistically robust by ignoring parts of the EEG signal that fell outside a standard deviation of 1.0 from the training signals.

The overall recognition results in Table 1 suggest that both the robust Kalman filter and ICA have a statistically significant advantage over correlation ($p < 0.01$). The robust Kalman filter has a very small advantage over ICA (not statistically significant).

In order to look at the reliability of the best algorithm and its ability to be used on-line two of the Subjects (S4 and S5) returned for another VR driving session. In these sessions the brakes of the driving simulator were controlled by the robust Kalman filter recognition algorithm for red stop and yellow slow down lights. Green lights were ignored. The results of this session using the Robust Kalman Filter trained on the first session are shown in Table 2. The recognition numbers for red and yellow lights between the two sessions were compared using correlation. Red light scores between the sessions correlated fairly highly - 0.82 for S4 and 0.69 for S5. The yellow light scores between sessions correlated poorly with both S4 and S5 at approximately -0.1. This indicates that the yellow light epochs tend to correlate poorly with each other due to the lack of a large component such as the P3 to tie them together.

# 4  Future Work

This paper showed the viability of recognizing the P3 EP in a VR environment. We plan to allow the P3 EP to propel the user through alternate virtual rooms through the use of various binary decisions. In order to improve recognition for the BCI we need to experiment with a wider and more complex variety of recognition algorithms. Our most recent work has shown a dependence between the human computer interface used in the BCI and recognition. We would like to explore this dependence in order to improve recognition as much as possible.

## Footnotes

*This research was supported by NIH/PHS grant1-P41-RR09283. It was also facilitated in part by a National Physical Science Consortium Fellowship and by stipend support from NASA Goddard Space Flight Center.

# References

[Bayliss and Ballard, 1998] J.D. Bayliss and D.H. Ballard, "The Effects of Eye Tracking in a VR Helmet on EEG Recording," *TR 685, University of Rochester National Resource Laboratory for the Study of Brain and Behavior*, May 1998.

[Chapman and Bragdon, 1964] R.M. Chapman and H.R. Bragdon, "Evoked responses to numerical and non-numerical visual stimuli while problem solving.," *Nature*, 203:1155–1157, 1964.

[Farwell and Donchin, 1988] L.A. Farwell and E. Donchin, "Talking off the top of your head: toward a mental prosthesis utilizing event-related brain potentials," *Electroenceph. Clin. Neurophysiol.*, pages 510–523, 1988.

[Jung *et al.*, 1997] T.P. Jung, C. Humphries, T. Lee, S. Makeig, M.J. McKeown, V. Iragui, and T.J. Sejnowski, "Extended ICA Removes Artifacts from Electroencephalographic Recordings," *to Appear in Advances in Neural Information Processing Systems*, 10, 1997.

[Makeig *et al.*, 1997] S. Makeig, T. Jung, A.J. Bell, D. Ghahremani, and T.J. Sejnowski, "Blind Separation of Auditory Event-related Brain Responses into Independent Components," *Proc. Nat'l Acad. Sci. USA*, 94:10979–10984, 1997.

[McFarland *et al.*, 1993] D.J. McFarland, G.W. Neat, R.F. Read, and J.R. Wolpaw, "An EEG-based method for graded cursor control," *Psychobiology*, 21(1):77–81, 1993.

[Pfurtscheller *et al.*, 1996] G. Pfurtscheller, D. Flotzinger, M. Pregenzer, J. Wolpaw, and D. McFarland, "EEG-based Brain Computer Interface (BCI)," *Medical Progress through Technology*, 21:111–121, 1996.

[Polich, 1998] J. Polich, "P300 Clinical Utility and Control of Variability," *J. of Clinical Neurophysiology*, 15(1):14–33, 1998.

[Rao, 1998] R. P.N. Rao, "Visual Attention during Recognition," *Advances in Neural Information Processing Systems*, 10, 1998.

[Rosvold *et al.*, 1956] H.E. Rosvold, A.F. Mirsky, I. Sarason, E.D. Bransome Jr., and L.H. Beck, "A Continuous Performance Test of Brain Damage," *J. Consult. Psychol.*, 20, 1956.

[Semlitsch *et al.*, 1986] H.V. Semlitsch, P. Anderer, P Schuster, and O. Presslich, "A solution for reliable and valid reduction of ocular artifacts applied to the P300 ERP," *Psychophys.*, 23:695–703, 1986.

[Sutton *et al.*, 1965] S. Sutton, M. Braren, J. Zublin, and E. John, "Evoked potential correlates of stimulus uncertainty," *Science*, 150:1187–1188, 1965.

[Vaughn *et al.*, 1996] T.M. Vaughn, J.R. Wolpaw, and E. Donchin, "EEG-Based Communication: Prospects and Problems," *IEEE Trans. on Rehabilitation Engineering*, 4(4):425–430, 1996.

[Vigario, 1997] R. Vigario, "Extraction of ocular artifacts from eeg using independent component analysis," *Electroenceph. Clin. Neurophysiol.*, 103:395–404, 1997.
